# Coarse-to-Fine Image Search Using Neural Networks

**Clay D. Spence, John C. Pearson, and Jim Bergen**
National Information Display Laboratory
P.O. Box 8619
Princeton, NJ 08543-8619
cspence@sarnoff.com
John_Pearson@maca.sarnoff.com
jbergen@sarnoff.com

## Abstract

The efficiency of image search can be greatly improved by using a coarse-to-fine search strategy with a multi-resolution image representation. However, if the resolution is so low that the objects have few distinguishing features, search becomes difficult. We show that the performance of search at such low resolutions can be improved by using *context* information, i.e., objects visible at low-resolution which are not the objects of interest but are associated with them. The networks can be given explicit context information as inputs, or they can learn to detect the context objects, in which case the user does not have to be aware of their existence. We also use *Integrated Feature Pyramids*, which represent high-frequency information at low resolutions. The use of multi-resolution search techniques allows us to combine information about the appearance of the objects on many scales in an efficient way. A natural form of exemplar selection also arises from these techniques. We illustrate these ideas by training hierarchical systems of neural networks to find clusters of buildings in aerial photographs of farmland.

## 1  INTRODUCTION

Coarse-to-fine image search is a general technique for improving the efficiency of any search algorithm. (Burt, 1988a and b; Burt et al., 1989). One begins by searching a low-resolution (coarse-scale) version of the image, typically obtained by constructing an image

pyramid (Burt and Adelson, 1983). Since this version is smaller than the original, there are far fewer pixels to be processed and the search is correspondingly faster. To improve the certainty of detection and refine the location estimates, the search process is repeated at higher resolution (finer scale), but only in those regions-of-interest (*ROIs*) which were identified at lower resolution as likely to contain one of the objects to be found, thus greatly reducing the actual area to be searched. This process is repeated at successively higher resolutions until sufficient certainty and precision is achieved or the original image has been searched.[1]

These pyramid techniques scale well with image size and can quickly process large images, but the relatively simple, hand-tuned pattern recognition components that are typically used limit their accuracy. Neural networks provide a complementary set of techniques, since they can learn complex patterns, but they don't scale well with image size. We have developed a Hybrid Pyramid/Neural Network (HPNN) system that combines the two techniques so as to leverage their strengths and compensate for their weaknesses.

A novel benefit of combining pyramids and neural networks is the potential for automatically learning to use *context* information, by which we mean any visible characteristics of the object's surroundings which tend to distinguish it from other objects. Examples of context information which might be useful in searching aerial photographs for man-made objects are the proximity of roads and terrain type. Pyramids provide the ability to detect context in a low-resolution image representation, which is necessary for efficient processing of large regions of the image. Neural networks provide the ability to discover context of which we may be unaware, and to learn the relevance of the context to detection, the dependence of detection probability on distance from a context object, etc. Context can help to narrow the search region at low resolutions and improve the detection performance since it is, by definition, relevant information. Of course, the usefulness of the idea will be different for each problem.

Context can be explicitly provided, e.g., in the form of a road map. As mentioned above, a neural net may also learn to exploit some context which is not explicitly provided, but must be inferred from the image data. If there is such context and the network's architecture and input features are capable of exploiting it, it should learn to do so even if we use ordinary training methods that do not explicitly take context into account.

We currently implement these functions in a hierarchical sequence of ordinary feed-forward networks with sigmoidal units, one network at each resolution in an image pyramid (Fig. 1). Each network receives some features extracted from a window in the image, and its output is interpreted as the probability of finding one of the searched-for objects in the center of the window. The network is scanned over all ROIs in this manner. (At the lowest resolution there is one ROI, which is the entire image.)

The networks are trained one at a time, starting at the lowest resolution. Context is made available to networks at higher resolutions by giving them information from the lower-resolution networks as additional inputs, which we will call *context inputs*. These could be

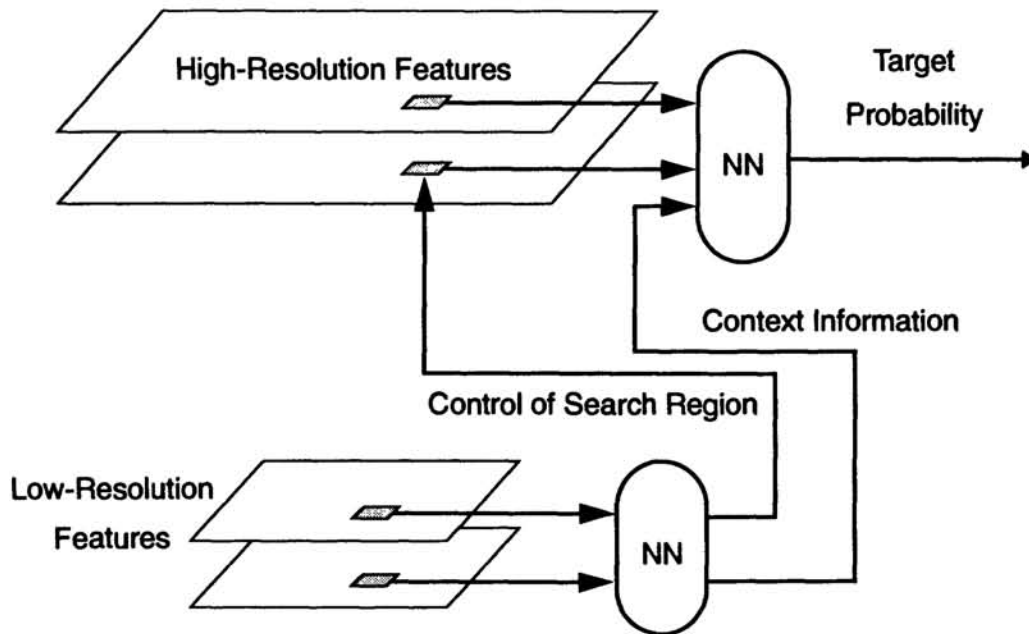

Figure 1. A Hierarchical Search System that Exploits Context.

taken from either the hidden units or output units of the lower-resolution networks. By training the networks in sequence starting at low resolution, we can choose to train the higher-resolution networks in the ROIs selected by the lower-resolution networks, thus providing a simple means of exemplar selection. This *coarse-to-fine training* is often useful, since many problems have relatively few positive examples per image, but because of the size of the image there are an enormous number of more-or-less redundant negative examples.

## 2    AN EXAMPLE PROBLEM

To demonstrate these ideas we applied them to the problem of finding clusters of buildings in aerial photographs of a rural area. Since buildings are almost always near a road, roads are context objects associated with buildings. We approached the problem in two ways to demonstrate different capabilities. First, we trained systems of networks with explicitly provided context in the form of road maps, in order to demonstrate the possible benefits.[2] Second, we trained systems without explicitly-provided context, in order to show that the context could be learned.

### 2.1    EXPLICIT CONTEXT

For comparison purposes, we trained one network to search a high-resolution image directly, with no explicit context inputs and no inputs from other networks. To demonstrate the effect of learned context on search we trained a second system with no explicit

context provided to the nets, but each received inputs from all of the networks at lower resolutions. These inputs were simply the outputs of those lower-resolution networks. To demonstrate the benefit of explicitly provided context, we trained a third system with both context inputs from lower-resolution networks and explicit context in the form of a road map of the area, which was registered with the image.

### 2.1.1 Features

To preserve some distinguishing features at a given low resolution, we extracted simple features at various resolutions and represented them at the lower resolution. The low-resolution representations were constructed by reducing the feature images with the usual blur-and-sub-sample procedure used to construct a Gaussian pyramid (Burt and Adelson, 1983). The features should not be too computationally expensive to extract, otherwise the efficiency benefit of coarse-to-fine search would be canceled.

The features used as inputs to the neural nets in the building-search systems were simple measures of the spatial image energy in several frequency bands. We constructed these feature images by building the Laplacian pyramid of the image[3] and then taking the absolute values of the pixels in each image in the pyramid. We then constructed a Gaussian pyramid of each of these images, to provide versions of them at different resolutions. A neural net searching a given resolution received input from each energy image derived from the same resolution and all higher resolutions.[4]

Binary road-map images were constructed from the digitized aerial photographs. They were reduced in resolution by first performing a binary blur of the image and then sub-sampling it by two in each dimension. In the binary blur procedure each pixel is set to one if any of its nearest neighbors were road pixels before blurring. This is repeated to get road maps at each resolution. To give the networks a rough measure of distance to a road, we gave the nets inputs from linearly-blurred versions of the road maps, which were made by expanding even lower-resolution versions of the road map with the same linear expand operation used in constructing the Laplacian pyramid. These blurred road maps are therefore not binary. The networks at the fifth and third pyramid levels received inputs from the road-maps at their resolution and from the two lower resolutions, while the network at the first pyramid level received input from the road map at its resolution and the next lower resolution.

### 2.1.2 Training the Networks

To estimate the probability of finding a building cluster at or around a given location in the image, each network received a single pixel from the same location in all of its input images. This should be adequate for search, since a single pixel in a feature image contains information about an extended region of the original. With the features we used, it also makes the system invariant to rotations, so the networks do not have to learn this invariance Also for simplicity, we did not train nets at all resolutions, but only at the fifth, third and first pyramid levels, i.e., on images which were one-thirty-second, one-eighth,

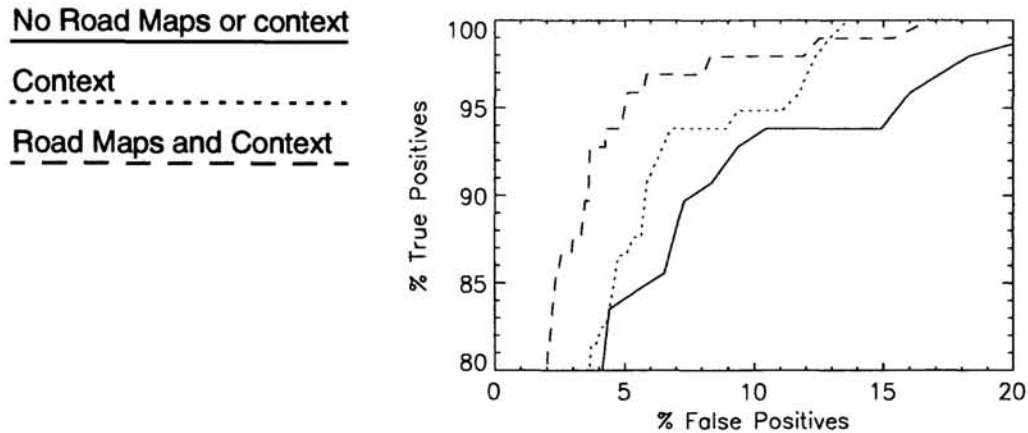

No Road Maps or context

Context

Road Maps and Context

Figure 2. ROC Curves for the Three Network Search Systems.

and one-half the width and height of the original image. A typical building cluster has a linear extent of about 30 pixels in the original images, so in the fifth Gaussian pyramid level they have few distinguishing features. The networks were the usual feed-forward nets with sum-and-sigmoid units and one hidden layer. They were trained using the cross-entropy error measure, with a desired output of one in a hand-chosen polygonal region about each building cluster. The standard quadratic weight-decay term was added to this to get the objective function, and only one regularization constant was used. This was adjusted to give lowest error on a test set. We often found that the weights to two or more hidden units would become identical and/or very small as the regularization term was increased, and in these cases we pruned the extraneous units and began the search for the optimal regularization parameter again. We usually ended up with very small networks, with from two to five hidden units. After training a net at low-resolution, we expanded the image of that net's output in order to use it as a context input for the networks at higher resolution.

### 2.1.3 Performance

To compare the performance of the three systems, we chose a threshold for the networks' outputs, and considered a building cluster to be detected by a network if the network's output was above the threshold at any pixel within the cluster. The number of detected clusters divided by the total number is the true-positive rate. The false-positive rate is more difficult to define, since we had in mind an application in which the detection system would draw a user's attention to objects which are likely to be of interest. The procedure we used attempts to measure the amount of effort a human would expend in searching regions with false detections by the network. See (Spence, et al., 1994) for details.

The performance figures presented here were measured on a validation set, i.e., an image on which the network was not trained and which was not used to set the regularization parameter. The results presented in Tables 1 and 2 are for a single threshold, chosen for each network so that the true-positive rate was 90%. Figure 2 compares the ROC curves of the three systems, i.e. the parametric curves of the true and false-positive rates as the threshold on the network's output is varied. From Table 1, the features we used would seem adequate for search at very low resolution, although the performance could be better.

Table 1: False-Positive Rates vs. Resolution.
These are results for the system with both road-map and context inputs.

| PYRAMID LEVEL | FALSE-POSITIVE RATE |
|---|---|
| 5 | 16% |
| 3 | 4.6% |
| 1 | 3.6% |

Table 2: False-Positive Rates of the Search Systems at 90% True-Positive Rate.

| NETWORK SYSTEM | FALSE-POSITIVE RATE |
|---|---|
| No Context or Road-Map Inputs | 7.6% |
| Context Inputs, No Road Maps | 5.8% |
| Context Inputs and Road Maps | 3.6% |

Table 2 and Figure 2 clearly show the benefits of using the road map and context inputs, although the statistics are somewhat poor at the higher true-positive rates because of the small number of building clusters which are being missed

## 2.2 LEARNING CONTEXT

Two things were changed to demonstrate that the context could be learned. First, the unoriented spatial energy features are not well suited for distinguishing between roads and other objects with a size similar to a road's width, so we used oriented energies instead. These were the oriented energies derived using steerable filters (Freeman and Adelson, 1991) at four orientations. To force orientation invariance, we sorted the four oriented energies at each pixel for each frequency band. These sorted oriented energy images were then reduced in size as appropriate for the resolution being searched by a network. In this case, we extracted energies only from the first, third, and fifth pyramid levels.

The second change is the use of hidden unit outputs for context inputs, instead of the output unit's outputs. The output units estimate the probability of finding a building cluster. Although this may reflect information about roads, it is very indirect information about the roads. It is more likely to carry some information about the coarse-scale appearance of the potential building clusters. The hidden unit outputs should contain a richer description of the image at a coarser scale.

### 2.2.1 Performance

The networks were trained in the same way as the networks described in Section 2.1. We trained three networks to search levels five, three, and one. For comparison purposes, we also trained a single-network to search in level one, with the same input features that were used for all of the networks of the hierarchical search system. These include, for example, three versions of each of the oriented energy images from level one (the second highest frequency band). Two of these versions were reduced in size to levels three and five, and then re-expanded to level one, so that they are simply blurred versions of the original energy images. This gives the network a direct source of information on the coarse-scale

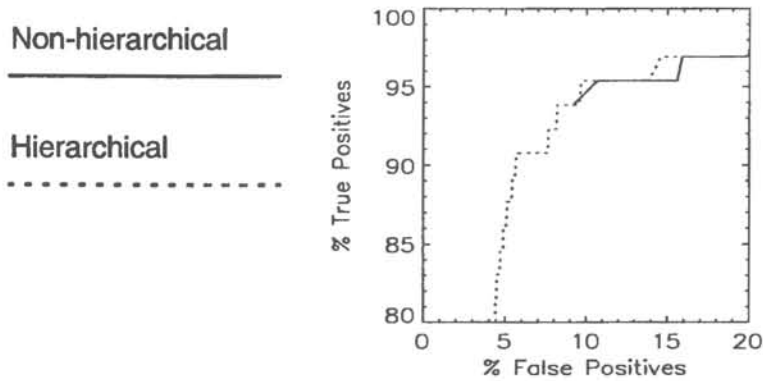

Figure 3. Performance of Hierarchical and Non-hierarchical Algorithms that Learn Context.

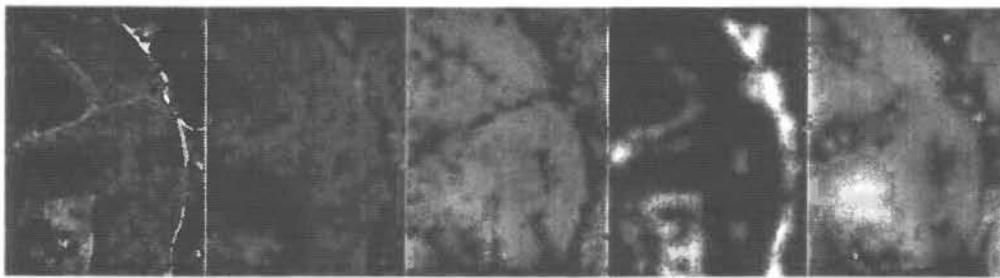

Figure 4. Image and Context Inputs to Highest-Resolution Network.

appearance of the image, although it is an expensive representation in terms of computation and memory, and the network has more inputs so that it takes longer to train.

The ROC curves for the two detection algorithms are shown in Figure 3. Their performance is about the same, suggesting that the coarser-scale networks in the hierarchical system are not performing computations that are useful for the finest-scale network, rather they simply pass on information about the coarse-scale appearance. In this example, the advantage of the hierarchical system is in the efficiency of both the training of the algorithm and its use after training. Figure 4 shows the level-one image and the outputs of several hidden units from the level-three network, expanded to the same size. These outputs suggest that road information is being extracted at coarse scale and passed to the high-resolution network.

## 3  DISCUSSION

We have performed "proof-of-concept" simulations on a realistic search problem of two of the key concepts of our Hybrid Pyramid/Neural Network (HPNN) approach. We have shown that: 1. The HPNN can learn to utilize explicitly-provided context data to improve object detection; 2. the networks can learn to use context without being explicitly taught to do so. This second point implies that the person training the system does not need to know of the existence of the context objects. The networks at the different scales are trained in sequence, starting at low resolution. A benefit of this *coarse-to-fine training* is a simple form of exemplar selection, since we can choose to train a network only in the regions of

interest as detected by the already-trained networks at lower resolution. We could also train all of the networks simultaneously, so that the lower-resolution networks learn to extract the most helpful information from the low-resolution image. The resulting system would probably perform better, but it would also be more expensive to train.

This approach should work quite well for many automatic target recognition problems, since the targets are frequently quite small. For more extended objects like clusters of buildings, our method is an efficient way of examining the objects on several length scales, but the detection at the finest scale is based on a small part of the potential object's appearance. Extended images of real objects typically have many features at each of several resolutions. We are currently working on techniques for discovering several characteristic features at each resolution by training several networks at each resolution, and integrating their responses to compute an overall probability of detection.

## Acknowledgments

We would like to thank Peter Burt, P. Anandan, and Paul Sajda for many helpful discussions. This was work was funded by the National Information Display Laboratory and under ARPA contract No. N00014-93-C-0202.

## Footnotes

[1] Several authors have investigated multi-scale processing of one-dimensional signals with neural networks, e.g., Mozer (1994) and Burr and Miyata (1993) studied music composition. Burr and Miyata use sub-sampling as in a pyramid decomposition. Images differ somewhat from music in that there are primitive features at all scales (limited by the sampling rate and image size), whereas the average frequency spectrum of music over a long time span doesn't seem likely to be meaningful. The original paper on pyramid image representation is (Burt and Adelson, 1983).

[2] It would be surprising if this extra information didn't help, since we know it is relevant. For many applications digitized maps will be available, so demonstrating the possible performance benefit is still worth-while.

[3] The Laplacian pyramid is usually constructed by expanding the lower-resolution levels of a Gaussian pyramid and subtracting each from the next-higher-resolution level. This gives a set of images which are band-passed with one-octave spacing between the bands.

[4] There are many examples of more sophisticated features, e.g., Lane, et al., 1992, Ballard and Wixson, 1993, and Greenspan, et al., 1994. Simple features were adequate for demonstrating the ideas of this paper.

## References

D.H. Ballard and L.E. Wixson (1993) Object recognition using steerable filters at multiple scales, *Proceedings of the IEEE Workshop on Qualitative Vision*, New York, NY.

D. Burr and Y. Miyata (1993) Hierarchical recurrent networks for learning musical structure, *Proceedings of the IEEE Conference on Neural Networks for Signal Processing III*, C. Kamm, G. Kuhn, B. Yoon, S.Y. Kung, and R. Chellappa, eds., Piscataway, NJ, pp. 216–225.

P. J. Burt and E. H. Adelson. (1983) The Laplacian pyramid as a compact image code. *IEEE Transactions*, Vol. COM-31:4, April, pp. 532–540.

P.J. Burt (1988a) Smart sensing with a pyramid vision machine, *Proceedings of the IEEE* Vol. 76, pp. 1006–1015.

P.J. Burt (1988b) Attention mechanisms for vision in a dynamic world, *Proceedings of the 9th International Conference on Pattern Recognition*, pp. 977–987.

P.J. Burt, J.R. Bergen, R. Kolczynski, R. Hingorani, W.A. Lee, A. Leung, J. Lubin, and H. Shvayster (1989) Object tracking with a moving camera, *Proceedings of the IEEE Workshop on Motion*, Irvine.

W.T. Freeman and E.H. Adelson (1991) The design and use of steerable filters, *IEEE Trans. PAMI*, **12**:9, pp. 891–906.

S.H. Lane, J.C. Pearson, and R. Sverdlove (1992) Neural networks for classifying image textures, *Proceedings of the Government Applications of Neural Networks Conference*, Dayton, Ohio.

M.C. Mozer (1994) Neural network music composition by prediction: Exploring the benefits of psychoacoustic constraints and multiscale processing, to appear in *Connection Science*.
